# Learning annotated hierarchies from relational data

**Daniel M. Roy, Charles Kemp, Vikash K. Mansinghka, and Joshua B. Tenenbaum**
CSAIL, Dept. of Brain & Cognitive Sciences, MIT, Cambridge, MA 02139
{droy, ckemp, vkm, jbt}@mit.edu

## Abstract

The objects in many real-world domains can be organized into hierarchies, where each internal node picks out a category of objects. Given a collection of features and relations defined over a set of objects, an *annotated* hierarchy includes a specification of the categories that are most useful for describing each individual feature and relation. We define a generative model for annotated hierarchies and the features and relations that they describe, and develop a Markov chain Monte Carlo scheme for learning annotated hierarchies. We show that our model discovers interpretable structure in several real-world data sets.

## 1 Introduction

Researchers in AI and cognitive science [1, 7] have proposed that hierarchies are useful for representing and reasoning about the objects in many real-world domains. One of the reasons that hierarchies are valuable is that they compactly specify categories at many levels of resolution, each node representing the category of objects at the leaves below the node. Consider, for example, the simple hierarchy shown in Figure 1a, which picks out five categories relevant to a typical university department: employees, staff, faculty, professors, and assistant professors.

Suppose that we are given a large data set describing the features of these employees and the interactions among these employees. Each of the five categories will account for some aspects of the data, but different categories will be needed for understanding different features and relations. "Faculty," for example, is the single most useful category for describing the employees that publish papers (Figure 1b), but three categories may be needed to describe the social interactions among the employees (Figure 1c). In order to understand the structure of the department, it is important not only to understand the hierarchical organization of the employees, but to understand which levels in the hierarchy are appropriate for describing each feature and each relation. Suppose, then, that an *annotated hierarchy* is a hierarchy along with a specification of the categories in the hierarchy that are relevant to each feature and relation.

The idea of an annotated hierarchy is one of the oldest proposals in cognitive science, and researchers including Collins and Quillian [1] and Keil [7] have argued that semantic knowledge is organized into representations with this form. Previous treatments of annotated hierarchies, however, usually suffer from two limitations. First, annotated hierarchies are usually hand-engineered, and there are few proposals describing how they might be learned from data. Second, annotated hierarchies typically capture knowledge only about the features of objects: relations between objects are rarely considered. We address both problems by defining a generative model for objects, features, relations, and hierarchies, and showing how it can be used to recover an annotated hierarchy from raw data.

Our generative model for feature data assumes that the objects are located at the leaves of a rooted tree, and that each feature is generated from a partition of the objects "consistent" with the hierarchy. A *tree-consistent partition* (henceforth, t-c partition) of the objects is a partition of the objects into disjoint categories, i.e. each class in the partition is exactly the set of leaves descending from some node in the tree. Therefore, a t-c partition can be uniquely encoded as the set of these nodes whose leaf descendants comprise the classes (Figure 1a,b). The simplest t-c partition is the singleton set

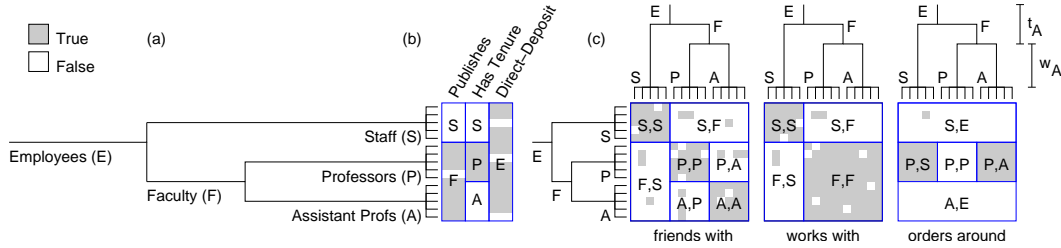

**Figure 1:** (a) A hierarchy over 15 members of a university department: 5 staff members, 5 professors and 5 assistant professors. (b) Three binary features, each of which is associated with a different t-c partition of the objects. Each class in each partition is labeled with the corresponding node in the tree. (c) Three binary relations, each of which is associated with a different t-c partition of the set of object pairs. Each class in each partition is labeled with the corresponding pair of nodes.

containing the root node, which places all objects into a single class. The most complex t-c partition is the set of all leaves, which assigns each object to its own class. We assume that the features of objects in different classes are independent, but that objects in the same class tend to have similar features. Therefore, finding the categories in the tree most relevant to a feature can be formalized as finding the simplest t-c partition that best accounts for the distribution of the feature (Figure 1b). We define an annotated hierarchy as a hierarchy together with a t-c partition for each feature.

Although most discussions of annotated hierarchies focus on features, much of the data available to human learners comes in the form of relations. Understanding the structure of social groups, for instance, involves inferences about relations like *admires*$(\cdot, \cdot)$, *friend-of*$(\cdot, \cdot)$ and *brother-of*$(\cdot, \cdot)$. Like the feature case, our generative model for relational data assumes that each (binary) relation is generated from a t-c partition of the set of all *pairs of objects*. Each class in a t-c partition now corresponds to a pair of categories (i.e. pair of nodes) (Figure 1c), and we assume that all pairs in a given class tend to take similar values. As in the feature case, finding the categories in the tree most relevant to a relation can be formalized as finding the t-c partition that best accounts for the distribution of the relation. The t-c partition for each relation can be viewed as an additional annotation of the tree. The final piece of our generative model is a prior over rooted trees representing hierarchies. Roughly speaking, the best hierarchy will then be the one that provides the best categories with which to summarize all the features and relations.

Like other methods for discovering structure in data, our approach may be useful both as a tool for data analysis and as a model of human learning. After describing our approach, we apply it to several data sets inspired by problems faced by human learners. Our first analysis suggests that the model recovers coherent domains given objects and features from several domains (animals, foods, tools and vehicles). Next we show that the model discovers interpretable structure in kinship data, and in data representing relationships between ontological kinds.

## 2   A generative model for features and relations

Our approach is organized around a generative model for feature data and relational data. For simplicity, we present our model for feature and relational data separately, focusing on the case where we have a single binary feature or a single binary relation. After presenting our generative model, we describe how it can be used to recover annotated hierarchies from data.

We begin with the case of a single binary feature and define a joint distribution over three entities: a rooted, weighted, binary tree $T$ with $\mathcal{O}$ objects at the leaves; a t-c partition of the objects; and feature observations, $\mathbf{d}$. For a feature, a t-c partition $\pi$ is a set of nodes $\{n_1, n_2, \ldots, n_k\}$, such that each object is a descendant of exactly one node in $\pi$. We will identify each node with the category of objects descending from it. We denote the data for all objects in the category $n$ as $\mathbf{d}_n$. If $o$ is a leaf (single object category), then $\mathbf{d}_o$ is the value of the feature for object $o$. In Figure 1b, three t-c partitions associated with the hierarchy are represented and each class in each partition is labeled with the corresponding category.

The joint distribution $P(T, w, \pi, \mathbf{d} | \lambda, \gamma_f)$ is induced by the following generative process:

i. Sample a tree $T$ from a uniform distribution over rooted binary trees with $\mathcal{O}$ leaves (each leaf will represent an object and there are $\mathcal{O}$ objects). Each node $n$ represents a category.

ii. For each category $n$, sample its weight, $w_n$, according to an exponential distribution with parameter $\lambda$, i.e. $p(w_n|\lambda) = \lambda e^{-\lambda w_n}$.

iii. Sample a t-c partition $\pi_f = \{n_1, n_2, \ldots, n_k\} \sim \Pi(\texttt{root-of}(T))$, where $\Pi(n)$ is a stochastic, set-valued function:

$$\Pi(n) = \begin{cases} \{n\} & n \text{ is a leaf, or w.p. } \phi(w_n) \\ \cup_i \Pi(n_i) & \text{otherwise} \end{cases} \qquad (1)$$

where $\phi(x) = 1 - e^{-x}$ and $n_i$ are the children of $n$. Intuitively, categories with large weight are more likely to be classes in the partition. For the *publishes* feature in Figure 1b, the t-c partition is $\{F, S\}$.

iv. For each category $n \in \pi_f$, sample $\theta_n \sim \text{Beta}(\gamma_f, \gamma_f)$, where $\theta_n$ is the probability that objects in category $n$ exhibit the feature $f$. Returning to the *publishes* example in Figure 1b, two parameters, $\theta_F$ and $\theta_S$, would be drawn for this feature.

v. For each object $o$, sample its feature value $\mathbf{d}_o \sim \text{Bernoulli}(\theta_n)$, where $n \in \pi_f$ is the category containing $o$.

Consider now the case where we have a single binary relation defined over all ordered pairs of objects $\{(o_i, o_j)\}$. In the relational case, our joint distribution is defined over a rooted, weighted, binary tree; a t-c partition of *ordered pairs* of objects; and observed, relational data represented as a matrix $\mathbf{D}$ where $\mathbf{D}_{i,j} = 1$ if the relation holds between $o_i$ and $o_j$.

Given a pair of categories $(n_i, m_j)$, let $n_i \times m_j$ be the set of all pairs of objects $(o_i, o_j)$ such that $o_i$ is an object in the category $n_i$ and $o_j$ is an object in the category $m_j$. With respect to pairs of trees, a t-c partition, $\pi$, is a set of pairs of categories $\{(n_1, m_1), (n_2, m_2), \ldots, (n_k, m_k)\}$ such that, for every pair of objects $(o_i, o_j)$, there exists exactly one pair $(n_k, m_k) \in \pi$ such that $(o_i, o_j) \in n_k \times m_k$. To help visualize these 2D t-c partitions, we can reorder the columns and rows of the matrix $\mathbf{D}$ according to an in-order traversal of the binary tree $T$. Each t-c partition now splits the matrix into contiguous, rectangular blocks (see Figure 1c, where each rectangular block is labeled with its category pair). Assuming we have already generated a rooted, weighted binary tree, we now specify the generative process for a single binary relation (c.f. steps iii through v in the feature case):

iii. Sample a t-c partition $\pi_r = \{(n_1, m_1), \ldots, (n_k, m_k)\} \sim \Pi(\texttt{root-of}(T), \texttt{root-of}(T))$, where $\Pi(n, m)$ is a stochastic, set-valued function:

$$\Pi(n, m) = \begin{cases} \{(n, m)\} & \text{w.p. } \phi(w_n) \cdot \phi(w_m) \\ \cup_i \Pi(n_i, m) & \text{otherwise, w.p. } \frac{1}{2} \\ \cup_j \Pi(n, m_j) & \text{otherwise} \end{cases} \qquad (2)$$

where $n_i/m_j$ are the children of $n/m$. To handle special cases, if both $n, m$ are leaves, $\Pi(n, m) = \{n, m\}$; if only one of the nodes is a leaf, we default to the feature case on the remaining tree, halting with probability $\phi(w_n) \cdot \phi(w_m)$. Intuitively, if a pair of categories $(n, m)$ both have large weight, the process is more likely to group all pairs of objects in $n \times m$ into a single class. In Figure 1c, the t-c partition for the *works with* relation is $\{(S, S), (S, F), (F, S), (F, F)\}$.

iv. For each pair of categories $(n, m) \in \pi_r$, sample $\theta_{n,m} \sim \text{Beta}(\gamma_r, \gamma_r)$, where $\theta_{n,m}$ is the probability that the relation holds between any pair of objects in $n \times m$. For the *works with* relation in Figure 1c, parameters would be drawn for each of the four classes in the t-c partition.

v. For each pair of objects $(o_i, o_j)$, sample the relation $\mathbf{D}_{i,j} \sim \text{Bernoulli}(\theta_{n,m})$, where $(n, m) \in \pi_r$ and $(o_i, o_j) \in (n, m)$. That is, the probability that the relation holds is the same for all pairs in a given class.

For data sets with multiple relations and features, we assume that all relations and features are conditionally independent given the weighted tree $T$.

## 2.1 Inference

Given observations of features and relations, we can use the generative model to ask various questions about the latent hierarchy and its annotations. We start by determining the posterior distribution on the weighted tree topologies, $(T, w)$, given data $D = (\{\mathbf{d}^{(f)}\}_{f=1}^{\mathcal{F}}, \{\mathbf{D}^{(r)}\}_{r=1}^{\mathcal{R}})$ over $\mathcal{O}$ objects, $\mathcal{F}$ features and $\mathcal{R}$ relations and hyperparameters $\lambda$ and $\gamma = (\{\gamma_f\}_{f=1}^{\mathcal{F}}, \{\gamma_r\}_{r=1}^{\mathcal{R}})$. By Bayes' rule,

$$P(T, w | D, \lambda, \gamma) \propto P(T) \quad P(w|T, \lambda) \qquad P(D|T, w, \gamma)$$

$$\propto \left(1\right) \quad \left(\prod_n \lambda e^{-\lambda w_n}\right) \quad \left(\prod_{f=1}^{\mathcal{F}} P(\mathbf{d}^{(f)}|T, w, \gamma_f) \prod_{r=1}^{\mathcal{R}} P(\mathbf{D}^{(r)}|T, w, \gamma_r)\right).$$

But $P(\mathbf{d}^{(f)}|T, w, \gamma_f) = \sum_\pi P(\pi|T, w) P(\mathbf{d}^{(f)}|\pi, \gamma_f)$, where $P(\pi|T, w)$ is the distribution over t-c partitions induced by the stochastic function $\Pi$ and $P(\mathbf{d}^{(f)}|\pi, \gamma_f)$ is the likelihood given the partition, marginalizing over the feature probabilities, $\theta_n$. Because the classes are independent, $P(\mathbf{d}^{(f)}|\pi, \gamma_f) = \prod_{n \in \pi} P(\mathbf{d}_n^{(f)}|n \in \pi, \gamma_f)$, where $\mathbb{M}_f(n) = P(\mathbf{d}_n^{(f)}|n \in \pi, \gamma_f)$ is the marginal likelihood for $\mathbf{d}_n^{(f)}$, the features for objects in category $n$. For our binary-valued data sets, $\mathbb{M}_f(n)$ is the standard marginal likelihood for the beta-binomial model. Because there are an exponential number of t-c partitions, we present an efficient dynamic program for calculating $\mathbb{T}_f(n) = P(\mathbf{d}_n^{(f)}|T, w, \gamma_f)$. Then, $\mathbb{T}_f(\texttt{root-of}(T)) = P(\mathbf{d}^{(f)}|T, w, \gamma_f)$ is the desired quantity.

First observe that, for all objects (i.e. leaf nodes) $o$, $\mathbb{T}_f(o) = \mathbb{M}_f(o)$. Let $n$ be a node and assume no ancestor of $n$ is in $\pi$. With probability $\phi(w_n) = 1 - e^{-w_n}$, category $n$ will be a single class and the contribution to $\mathbb{T}_f$ will be $\mathbb{M}_f(n)$. Otherwise, $\Pi(n)$ splits category $n$ into its children, $n_1$ and $n_2$. Now the possible partitions of the objects in category $n$ are every t-c partition of the objects below $n_1$ paired with every t-c partition below $n_2$. By independence, this contributes $\mathbb{T}_f(n_1)\mathbb{T}_f(n_2)$. Hence,

$$\mathbb{T}_f(n) = \begin{cases} \phi(w_n)\mathbb{M}_f(n) + (1 - \phi(w_n))\,\mathbb{T}_f(n_1)\mathbb{T}_f(n_2) & \text{if } n \text{ is an internal node} \\ \mathbb{M}_f(n) & \text{otherwise.} \end{cases}$$

For the relational case, we describe a dynamic program $\mathbb{T}_r(n, m)$ that calculates $P(\mathbf{D}_{n,m}^{(r)}|T, w, \gamma_r)$, the probability of all relations between objects in $n \times m$, conditioned on the tree, having marginalized out the t-c partitions and relation probabilities. Let $\mathbb{M}_r(n, m) = P(\mathbf{D}_{n,m}^{(r)}|(n, m) \in \pi, \gamma_r)$ be the marginal likelihood of the relations in $n \times m$. For relations, $\mathbb{M}_f(n, m)$ is also the beta-binomial. If $n$ and $m$ are both leaves, then $\mathbb{T}_r(n, m) = \mathbb{M}_r(n, m)$. Otherwise,

$$\mathbb{T}_r(n, m) = \phi(w_n)\,\phi(w_m)\mathbb{M}_r(n, m)$$

$$+ (1 - \phi(w_n)\,\phi(w_m)) \begin{cases} \mathbb{T}_r(n, m_1)\mathbb{T}_r(n, m_2) & n \text{ is a leaf} \\ (\mathbb{T}_r(n_1, m)\mathbb{T}_r(n_2, m) & m \text{ is a leaf} \\ \frac{1}{2} \cdot (\mathbb{T}_r(n, m_1)\mathbb{T}_r(n, m_2) + \mathbb{T}_r(n_1, m)\mathbb{T}_r(n_2, m)) & \text{otherwise} \end{cases}$$

The above dynamic programs have linear and quadratic complexity in the number of objects, respectively. Because we can efficiently compute the posterior density of a weighted tree, we can search for the maximum *a posteriori* (MAP) weighted tree. Conditioned on the MAP tree, we can efficiently compute the MAP t-c partition for each feature and relation. We find the MAP tree first, rather than jointly optimizing for both the topology and partitions, because marginalizing over the t-c partitions produces more robust trees; marginalization has a (Bayesian) "Occam's razor" effect and helps avoid overfitting. MAP t-c partitions can be computed by a straightforward modification of the above dynamic programs, replacing sums with max operations and maintaining a list of nodes representing the MAP t-c partition at each node in the tree.

We chose to implement global search by building a Markov chain Monte Carlo (MCMC) algorithm with the posterior as the stationary distribution and keeping track of the best tree as the chain mixes. For all the results in this paper, we fixed the hyperparameters of all beta distributions to $\gamma = 0.5$ (i.e. the asymptotically least informative prior) and report the (empirical) MAP tree and MAP t-c partitions conditioned on the tree. The MCMC algorithm searches for the MAP tree by cycling through three Metropolis-Hastings (MH) moves adapted from [14]:

    i. Subtree Pruning and Regrafting: Choose a node $n$ uniformly at random (except the root). Choose a non-descendant node $m$. Detach $n$ from its parent and collapse the parent (remove

node, attaching the remaining child to the parent's parent and adding the parent's weight to the child's). Sample $u \sim \text{Uniform}(0, 1)$ and then insert a new node $m'$ between $m$ and its parent. Attach $n$ to $m'$, set $w_{m'} := (1 - u)w_m$ and set $w_m := uw_m$.

ii. Edge Weight Adjustment: Choose a node $n$ uniformly at random (including the root) and propose a new weight $w_n$ (e.g. let $x$ be $\text{Normal}(\log(w_t), 1)$ and let new weight be $e^x$).

iii. Subtree Swapping: Choose a node $n$ uniformly at random (except the root). Choose another node $n'$ such that neither $n$ nor $n'$ is a descendant of the other, and swap $n$ and $n'$.

The first two moves suffice to make the chain ergodic; subtree swapping is included to improve mixing. The first and last moves are symmetric. We initialized the chain on a random tree with weights set to one, ran the chain for approximately one million iterations and assessed convergence by comparing separate chains started from multiple random initial states.

## 2.2 Related Work

There are several methods that discover hierarchical structure in feature data. Hierarchical clustering [4] has been successfully used for analyzing both biological data [18] and psychological data, but cannot learn the annotated hierarchies that we consider. Bayesian hierarchical clustering (BHC) [6] is a recent alternative which constructs a tree as a byproduct of approximate inference in a flat clustering model, but lacks any notion of annotations. It is possible that a BHC-inspired algorithm could be derived to find approximate MAP annotated hierarchies. Our model for feature data is most closely related to methods for Bayesian phylogenetics [14]. These methods typically assume that features are generated directly by a stochastic process over a tree. Our model adds an intervening layer of abstraction by assuming that partitions are generated by a stochastic process over a tree, and that features are generated from these partitions. By introducing a partition for each feature, we gain the ability to annotate a hierarchy with the levels most relevant to each feature.

There are several methods for discovering hierarchical structure in relational data [5, 13], but none of these methods provides a general purpose solution to the problem we consider. Most of these methods take a single relation as input, and assume that the hierarchy captures an underlying community structure: in other words, objects that are often paired in the input are assumed to lie nearby in the tree. Our approach handles multiple relations simultaneously, and allows a more flexible mapping between each relation and the underlying hierarchy. Different relations may depend on very different regions of the hierarchy, and some relations may establish connections between categories that are quite distant in the tree (see Figure 4).

Many non-hierarchical methods for relational clustering have also been developed [10, 16, 17]. One family of approaches is based on the stochastic blockmodel [15], of which the Infinite Relational Model (IRM) [9] is perhaps the most flexible. The IRM handles multiple relations simultaneously, and does not assume that each relation has underlying community structure. The IRM, however, does not discover hierarchical structure; instead it partitions the objects into a set of non-overlapping categories. Our relational model is an extension of the blockmodel that discovers a nested set of categories as well as which categories are useful for understanding each relation in the data set.

## 3 Results

We applied our model to three problems inspired by tasks that human learners are required to solve. Our first application used data collected in a feature-listing task by Cree and McRae [2]. Participants in this task listed the features that came to mind when they thought about a given object: when asked to think about a lemon, for example, subjects listed features like "yellow," "sour," and "grows on trees."[1] We analyzed a subset of the full data set including 60 common objects and the 100 features most commonly listed for these objects. The 60 objects are shown in Figure 2, and were chosen to represent four domains: animals, food, vehicles and tools.

Figure 2 shows the MAP tree identified by our algorithm. The model discovers the four domains as well as superordinate categories (e.g. "living things", including fruits, vegetables, and animals) and subordinate categories (e.g. "wheeled vehicles"). Figure 2 also shows MAP partitions for 10

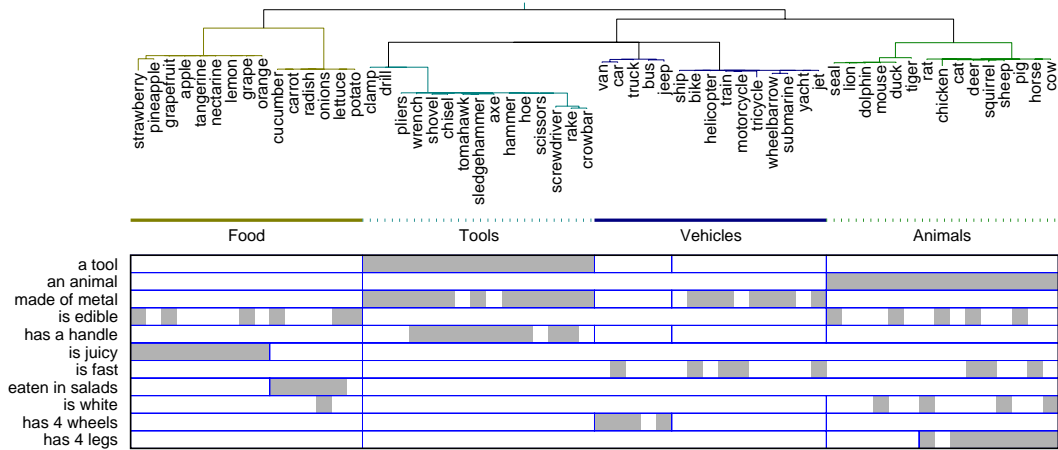

**Figure 2:** MAP tree recovered from a data set including 60 objects from four domains. MAP partitions for several features are shown: the model discovers, for example, that "is juicy" is associated with only one part of the tree. The weight of each edge in the tree is proportional to its vertical extent.

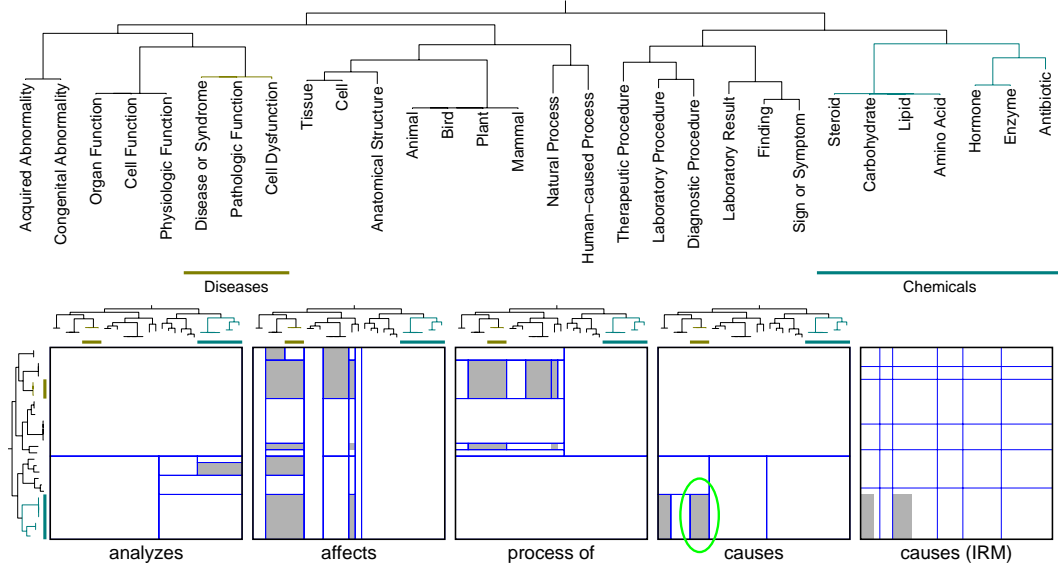

**Figure 3:** MAP tree recovered from 49 relations between entities in a biomedical data set. Four relations are shown (rows and columns permuted to match in-order traversal of the MAP tree). Consider the circled subset of the t-c partition for *causes*. This block captures the knowledge that "chemicals" *cause* "diseases." The Infinite Relational Model (IRM) does not capture the appropriate structure in the relation *cause* because it does not model the latent hierarchy, instead choosing a single partition to describe the structure across *all* relations.

representative features. The model discovers that some features are associated only with certain parts of the tree: "is juicy" is associated with the fruits, and "is metal" is associated with the man-made items. Discovering domains is a fundamental cognitive problem that may be solved early in development [11], but that is ignored by many cognitive models, which consider only carefully chosen data from a single domain (e.g. data including only animals and only biological features). By organizing the 60 objects into domains and identifying a subset of features that are associated with each domain, our model begins to suggest how infants may parse their environment into coherent domains of objects and features.

Our second application explores the acquisition of ontological knowledge, a problem that has been previously discussed by Keil [7]. We demonstrate that our model discovers a simple biomedical ontology given data from the Unified Medical Language System (UMLS) [12]. The full data set includes 135 entities and 49 binary relations, where the entities are ontological categories like 'Sign or Symptom', 'Cell', and 'Disease or Syndrome,' and the relations include verbs like *causes*, *analyzes* and *affects*. We applied our model to a subset of the data including the 30 entities shown in Figure 3.

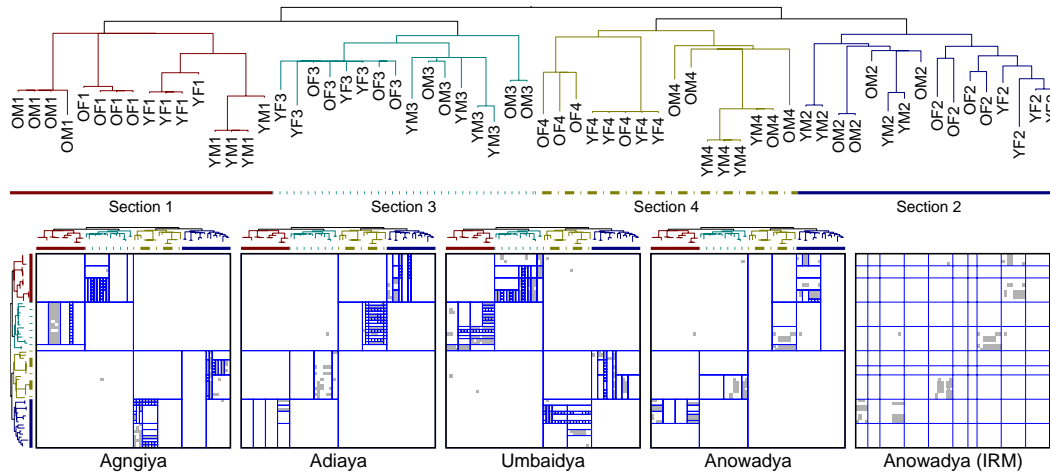

**Figure 4:** MAP tree recovered from kinship relations between 64 members of the Alyawarra tribe. Individuals have been labelled with their age, gender and kinship section (e.g. "YF1" is a young female from section 1). MAP partitions are shown for four representative relations: the model discovers that different relations depend on the tree in very different ways; hierarchical structure allows for a compact representation (c.f. IRM).

The MAP tree is an ontology that captures several natural groupings, including a category for "living things" (plant, bird, animal and mammal), a category for "chemical substances" (amino acid, lipid, antibiotic, enzyme etc.) and a category for abnormalities. The MAP partitions for each relation identify the relevant categories in the tree relatively cleanly: the model discovers, for example, that the distinction between "living things" and "abnormalities" is irrelevant to the first place of the relation *causes*, since neither of these categories can cause anything (according to the data set). This distinction, however, is relevant to the second place of *causes*: substances can cause abnormalities and dysfunctions, but cannot *cause* "living things". Note that the MAP partitions for *causes* and *analyzes* are rather different: one of the reasons why discovering separate t-c partitions for each relation is important is that different relations can depend on very different parts of an ontology.

Our third application is inspired by the problem children face when learning the kinship structure of their social group. This problem is especially acute for children growing up in Australian tribes, which have kinship systems that are more complicated in many ways than Western kinship systems, but which nevertheless display some striking regularities. We focus here on data from the Alyawarra tribe [3]. Denham [3] collected a large data set by asking 104 tribe members to provide kinship terms for each other. Twenty-six different terms were mentioned in total, and four of them are represented in Figure 4. More than one kinship term may describe the relationship between a pair of individuals — since the data set includes only one term per pair, some of the zeros in each matrix represent missing data rather than relationships that do not hold. For simplicity, however, we assume that relationships that were never mentioned do not exist.

The Alyawarra tribe is divided into four kinship sections, and these sections are fundamental to the social structure of the tribe. Each individual, for instance, is permitted only to marry individuals from one of the other sections. Whether a kinship term applies between a pair of individuals depends on their sections, ages and genders [3, 8]. We analyzed a subset of the full data set including 64 individuals chosen to equally represent all four sections, both genders, and people young and old. The MAP tree divides the individuals perfectly according to kinship section, and discovers additional structure within each section. Group three, for example, is split by age and then by gender. The MAP partitions for each relation indicate that different relations depend very differently on the structure of the tree. *Adiadya* refers to a younger member of one's own kinship section. The MAP partition for this relation contains fine-level structure only along the diagonal, indicating that the model has discovered that the term only applies between individuals from the same kinship section. *Umbaidya* can be used only between members of sections 1 and 3, and members of sections 2 and 4. Again the MAP partition indicates that the model has discovered this structure. In some places the MAP partitions appears to overfit the data: the partition for *Umbaidya*, for example, appears to capture some of the noise in this relation. This result may reflect the fact that our generative process is not quite right for these data: in particular, it does not capture the idea that some of the zeroes in each relation represent missing data.

# 4 Conclusions

We developed a probabilistic model that assumes that features and relations are generated over an annotated hierarchy, and showed how this model can be used to recover annotated hierarchies from raw data. Three applications of the model suggested that it is able to recover interpretable structure in real-world data, and may help to explain the computational principles which allow human learners to acquire hierarchical representations of real-world domains.

Our approach opens up several avenues for future work. A hierarchy specifies a set of categories, and annotations indicate which of these categories are important for understanding specific features and relations. A natural extension is to learn sets of categories that possess other kinds of structure, such as factorial structure [17]. For example, the kinship data we analyzed may be well described by three sets of overlapping categories where each individual belongs to a kinship section, a gender, and an age group. We have already extended the model to handle continuous data and can imagine other extensions, including higher-order relations, multiple trees, and relations between distinct sets of objects (e.g. given information, say, about the book-buying habits of a set of customers, this extension of our model could discover a hierarchical representation of the customers and a hierarchical representation of the books, and discover the categories of books that tend to be preferred by different kinds of customers). We are also actively exploring variants of our model that permit accurate online approximations for inference; e.g., by placing an exchangeable prior over tree structures based on a Polya-urn scheme, we can derive an efficient particle filter.

We have shown that formalizing the intuition behind annotated hierarchies in terms of a *prior* on trees and partitions and a *noise-robust likelihood* enabled us to discover interesting structure in real-world data. We expect a fruitful area of research going forward will involve similar marriages between *intuitions about structured representation* from classical AI and cognitive science and *modern inferential machinery* from Bayesian statistics and machine learning.

## Footnotes

[1]Note that some of the features are noisy — according to these data, onions are not edible, since none of the participants chose to list this feature for onion.

# References

[1] A. M. Collins and M. R. Quillian. Retrieval Time from Semantic Memory. *JVLVB*, 8:240–248, 1969.

[2] G. Cree and K. McRae. Analyzing the factors underlying the structure and computation of the meaning of chipmunk, chisel, cheese, and cello (and many other concrete nouns). *JEP Gen.*, 132:163–201, 2003.

[3] W. Denham. *The detection of patterns in Alyawarra nonverbal behaviour*. PhD thesis, U. of Wash., 1973.

[4] R. O. Duda and P. E. Hart. *Pattern Classification and Scene Analaysis*. Wiley, 2001.

[5] M. Girvan and M. E. J. Newman. Community structure in social and biological networks. *Proceedings of the National Academy of Sciences*, 99(12):7821–7826, 2002.

[6] K. Heller and Z. Ghahramani. Bayesian Hierarchical Clustering. In *ICML*, 2005.

[7] F. C. Keil. *Semantic and Conceptual Development*. Harvard University Press, Cambridge, MA, 1979.

[8] C. Kemp, T. L. Griffiths, and J. B. Tenenbaum. Discovering Latent Classes in Relational Data. Technical Report AI Memo 2004-019, MIT, 2004.

[9] C. Kemp, J. B. Tenenbaum, T. L. Griffiths, T. Yamada, and N. Ueda. Learning systems of concepts with an infinite relational model. In *AAAI*, 2006.

[10] J. Kubica, A. Moore, J. Schneider, and Y. Yang. Stochastic link and group detection. In *NCAI*, 2002.

[11] J. M. Mandler and L. McDonough. Concept formation in infancy. *Cog. Devel.*, 8:291–318, 1993.

[12] A. T. McCray. An upper level ontology for the biomedical domain. *Comp. Func. Genom.*, 4:80–84, 2001.

[13] J. Neville, M. Adler, and D. Jensen. Clustering relational data using attribute and link information. In *Proc. of the Text Mining and Link Analysis Workshop, IJCAI*, 2003.

[14] D. L. Swofford, G. J. Olsen, P. J. Waddell, and D. M. Hillis. Phylogenetic inference. *Molecular Systematics, 2nd. edition*, 1996.

[15] Y. J. Wang and G. Y. Wong. Stochastic blockmodels for directed graphs. *JASA*, 82:8–19, 1987.

[16] S. Wasserman and K. Faust. *Social network analysis: Methods and applications*. Cambridge Press, 1994.

[17] A. P. Wolfe and D. Jensen. Playing multiple roles: discovering overlapping roles in social networks. In *Proc. of the Workshop on statistical relational learning and its connections to other fields, ICML*, 2004.

[18] K. Y. Yeung, M. Medvedovic, and R. E. Bumgarner. Clustering gene-expression data with repeated measurements. *Genome Biology*, 2003.
